# Optimal Brain Damage

Yann Le Cun, John S. Denker and Sara A. Solla
AT&T Bell Laboratories, Holmdel, N. J. 07733

## ABSTRACT

We have used information-theoretic ideas to derive a class of practical and nearly optimal schemes for adapting the size of a neural network. By removing unimportant weights from a network, several improvements can be expected: better generalization, fewer training examples required, and improved speed of learning and/or classification. The basic idea is to use second-derivative information to make a tradeoff between network complexity and training set error. Experiments confirm the usefulness of the methods on a real-world application.

## 1    INTRODUCTION

Most successful applications of neural network learning to real-world problems have been achieved using highly structured networks of rather large size [for example (Waibel, 1989; Le Cun et al., 1990a)]. As applications become more complex, the networks will presumably become even larger and more structured. Design tools and techniques for comparing different architectures and minimizing the network size will be needed. More importantly, as the number of parameters in the systems increases, overfitting problems may arise, with devastating effects on the generalization performance. We introduce a new technique called Optimal Brain Damage (OBD) for reducing the size of a learning network by selectively deleting weights. We show that OBD can be used both as an automatic network minimization procedure and as an interactive tool to suggest better architectures.

The basic idea of OBD is that it is possible to take a perfectly reasonable network, delete half (or more) of the weights and wind up with a network that works just as well, or better. It can be applied in situations where a complicated problem must

be solved, and the system must make optimal use of a limited amount of training data. It is known from theory (Denker et al., 1987; Baum and Haussler, 1989; Solla et al., 1990) and experience (Le Cun, 1989) that, for a fixed amount of training data, networks with too many weights do not generalize well. On the other hand. networks with too few weights will not have enough power to represent the data accurately. The best generalization is obtained by trading off the training error and the network complexity.

One technique to reach this tradeoff is to minimize a cost function composed of two terms: the ordinary training error, plus some measure of the network complexity. Several such schemes have been proposed in the statistical inference literature [see (Akaike, 1986; Rissanen, 1989; Vapnik, 1989) and references therein] as well as in the NN literature (Rumelhart, 1988; Chauvin, 1989; Hanson and Pratt, 1989; Mozer and Smolensky, 1989).

Various complexity measures have been proposed, including Vapnik-Chervonenkis dimensionality (Vapnik and Chervonenkis, 1971) and description length (Rissanen, 1989). A time-honored (albeit inexact) measure of complexity is simply the number of non-zero free parameters, which is the measure we choose to use in this paper [but see (Denker, Le Cun and Solla, 1990)]. Free parameters are used rather than connections, since in constrained networks, several connections can be controlled by a single parameter.

In most cases in the statistical inference literature, there is some *a priori* or heuristic information that dictates the order in which parameters should be deleted; for example, in a family of polynomials, a smoothness heuristic may require high-order terms to be deleted first. In a neural network, however, it is not at all obvious in which order the parameters should be deleted.

A simple strategy consists in deleting parameters with small "saliency", i.e. those whose deletion will have the least effect on the training error. Other things being equal, small-magnitude parameters will have the least saliency, so a reasonable initial strategy is to train the network and delete small-magnitude parameters in order. After deletion, the network should be retrained. Of course this procedure can be iterated; in the limit it reduces to continuous weight-decay during training (using disproportionately rapid decay of small-magnitude parameters). In fact, several network minimization schemes have been implemented using non-proportional weight decay (Rumelhart, 1988; Chauvin, 1989; Hanson and Pratt, 1989), or "gating coefficients" (Mozer and Smolensky, 1989). Generalization performance has been reported to increase significantly on the somewhat small problems examined. Two drawbacks of these techniques are that they require fine-tuning of the "pruning" coefficients to avoid catastrophic effects, and also that the learning process is significantly slowed down. Such methods include the implicit hypothesis that the appropriate measure of network complexity is the number of parameters (or sometimes the number of units) in the network.

One of the main points of this paper is to move beyond the approximation that "magnitude equals saliency", and propose a theoretically justified saliency measure.

Our technique uses the second derivative of the objective function with respect to the parameters to compute the saliencies. The method was validated using our handwritten digit recognition network trained with backpropagation (Le Cun et al., 1990b).

## 2    OPTIMAL BRAIN DAMAGE

Objective functions play a central role in this field; therefore it is more than reasonable to define the saliency of a parameter to be the change in the objective function caused by deleting that parameter. It would be prohibitively laborious to evaluate the saliency directly from this definition, i.e. by temporarily deleting each parameter and reevaluating the objective function.

Fortunately, it is possible to construct a local model of the error function and *analytically predict* the effect of perturbing the parameter vector. We approximate the objective function $E$ by a Taylor series. A perturbation $\delta U$ of the parameter vector will change the objective function by

$$\delta E = \sum_i g_i \delta u_i + \frac{1}{2} \sum_i h_{ii} \delta u_i^2 + \frac{1}{2} \sum_{i \neq j} h_{ij} \delta u_i \delta u_j + O(||\delta U||^3) \tag{1}$$

Here, the $\delta u_i$'s are the components of $\delta U$, the $g_i$'s are the components of the gradient $G$ of $E$ with respect to $U$, and the $h_{ij}$'s are the elements of the Hessian matrix $H$ of $E$ with respect to $U$:

$$g_i = \frac{\partial E}{\partial u_i} \quad \text{and} \quad h_{ij} = \frac{\partial^2 E}{\partial u_i \partial u_j} \tag{2}$$

The goal is to find a set of parameters whose deletion will cause the least increase of $E$. This problem is practically insoluble in the general case. One reason is that the matrix $H$ is enormous ($6.5 \times 10^6$ terms for our 2600 parameter network), and is very difficult to compute. Therefore we must introduce some simplifying approximations. The "diagonal" approximation assumes that the $\delta E$ caused by deleting several parameters is the sum of the $\delta E$'s caused by deleting each parameter individually; cross terms are neglected, so third term of the right hand side of equation 1 is discarded. The "extremal" approximation assumes that parameter deletion will be performed after training has converged. The parameter vector is then at a (local) minimum of $E$ and the first term of the right hand side of equation 1 can be neglected. Furthermore, at a local minimum, all the $h_{ii}$'s are non-negative, so any perturbation of the parameters will cause $E$ to increase or stay the same. Thirdly, the "quadratic" approximation assumes that the cost function is nearly quadratic so that the last term in the equation can be neglected. Equation 1 then reduces to

$$\delta E = \frac{1}{2} \sum_i h_{ii} \delta u_i^2 \tag{3}$$

## 2.1  COMPUTING THE SECOND DERIVATIVES

Now we need an efficient way of computing the diagonal second derivatives $h_{ii}$. Such a procedure was derived in (Le Cun, 1987), and was the basis of a fast back-propagation method used extensively in various applications (Becker and Le Cun, 1989; Le Cun, 1989; Le Cun et al., 1990a). The procedure is very similar to the back-propagation algorithm used for computing the first derivatives. We will only outline the procedure; details can be found in the references.

We assume the objective function is the usual mean-squared error (MSE); generalization to other additive error measures is straightforward. The following expressions apply to a single input pattern; afterward $E$ and $H$ must be averaged over the training set. The network state is computed using the standard formulae

$$x_i = f(a_i) \quad \text{and} \quad a_i = \sum_j w_{ij} x_j \tag{4}$$

where $x_i$ is the state of unit $i$, $a_i$ its total input (weighted sum), $f$ the squashing function and $w_{ij}$ is the connection going from unit $j$ to unit $i$. In a shared-weight network like ours, a single parameter $u_k$ can control one or more connections: $w_{ij} = u_k$ for all $(i, j) \in V_k$, where $V_k$ is a set of index pairs. By the chain rule, the diagonal terms of $H$ are given by

$$h_{kk} = \sum_{(i,j) \in V_k} \frac{\partial^2 E}{\partial w_{ij}^2} \tag{5}$$

The summand can be expanded (using the basic network equations 4) as:

$$\frac{\partial^2 E}{\partial w_{ij}^2} = \frac{\partial^2 E}{\partial a_i^2} x_j^2 \tag{6}$$

The second derivatives are back-propagated from layer to layer:

$$\frac{\partial^2 E}{\partial a_i^2} = f'(a_i)^2 \sum_l w_{li}^2 \frac{\partial^2 E}{\partial a_l^2} - f''(a_i) \frac{\partial E}{\partial x_i} \tag{7}$$

We also need the boundary condition at the output layer, specifying the second derivative of $E$ with respect to the last-layer weighted sums:

$$\frac{\partial^2 E}{\partial a_i^2} = 2f'(a_i)^2 - 2(d_i - x_i)f''(a_i) \tag{8}$$

for all units $i$ in the output layer.

As can be seen, computing the diagonal Hessian is of the same order of complexity as computing the gradient. In some cases, the second term of the right hand side of the last two equations (involving the second derivative of $f$) can be neglected. This corresponds to the well-known Levenberg-Marquardt approximation, and has the interesting property of giving guaranteed positive estimates of the second derivative.

## 2.2  THE RECIPE

The OBD procedure can be carried out as follows:

1. Choose a reasonable network architecture
2. Train the network until a reasonable solution is obtained
3. Compute the second derivatives $h_{kk}$ for each parameter
4. Compute the saliencies for each parameter: $s_k = h_{kk} u_k^2/2$
5. Sort the parameters by saliency and delete some low-saliency parameters
6. Iterate to step 2

Deleting a parameter is defined as setting it to 0 and freezing it there. Several variants of the procedure can be devised, such as decreasing the values of the low-saliency parameters instead of simply setting them to 0, or allowing the deleted parameters to adapt again after they have been set to 0.

## 2.3  EXPERIMENTS

The simulation results given in this section were obtained using back-propagation applied to handwritten digit recognition. The initial network was highly constrained and sparsely connected, having $10^5$ connections controlled by 2578 free parameters. It was trained on a database of segmented handwritten zipcode digits and printed digits containing approximately 9300 training examples and 3350 test examples. More details can be obtained from the companion paper (Le Cun et al., 1990b).

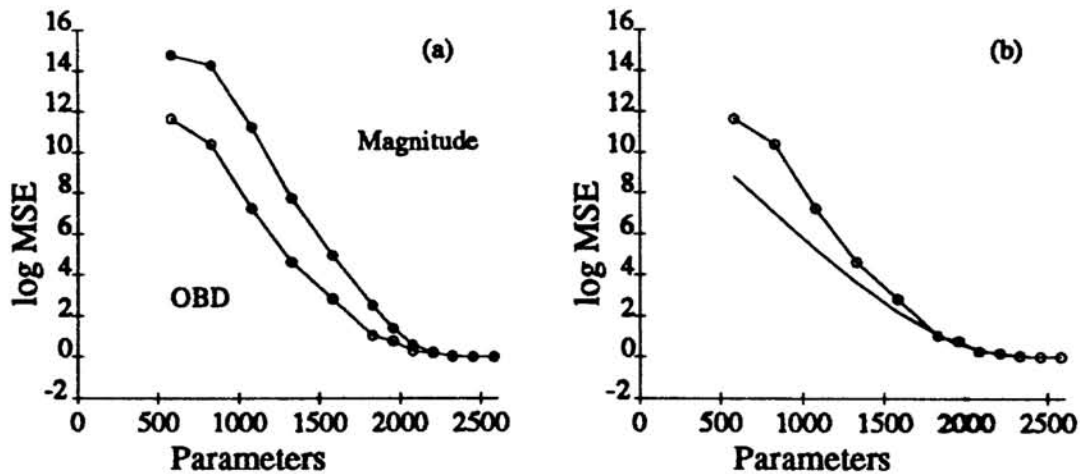

**Figure 1:** (a) Objective function (in dB) versus number of parameters for OBD (lower curve) and magnitude-based parameter deletion (upper curve). (b) Predicted and actual objective function versus number of parameters. The predicted value (lower curve) is the sum of the saliencies of the deleted parameters.

Figure 1a shows how the objective function increases (from right to left) as the number of remaining parameters decreases. It is clear that deleting parameters by

order of saliency causes a significantly smaller increase of the objective function than deleting them according to their magnitude. Random deletions were also tested for the sake of comparison, but the performance was so bad that the curves cannot be shown on the same scale.

Figure 1b shows how the objective function increases (from right to left) as the number of remaining parameters decreases, compared to the increase predicted by the Quadratic-Extremum-Diagonal approximation. Good agrement is obtained for up to approximately 800 deleted parameters (approximately 30% of the parameters). Beyond that point, the curves begin to split, for several reasons: the off-diagonal terms in equation 1 become disproportionately more important as the *number* of deleted parameters increases, and higher-than-quadratic terms become more important when *larger-valued* parameters are deleted.

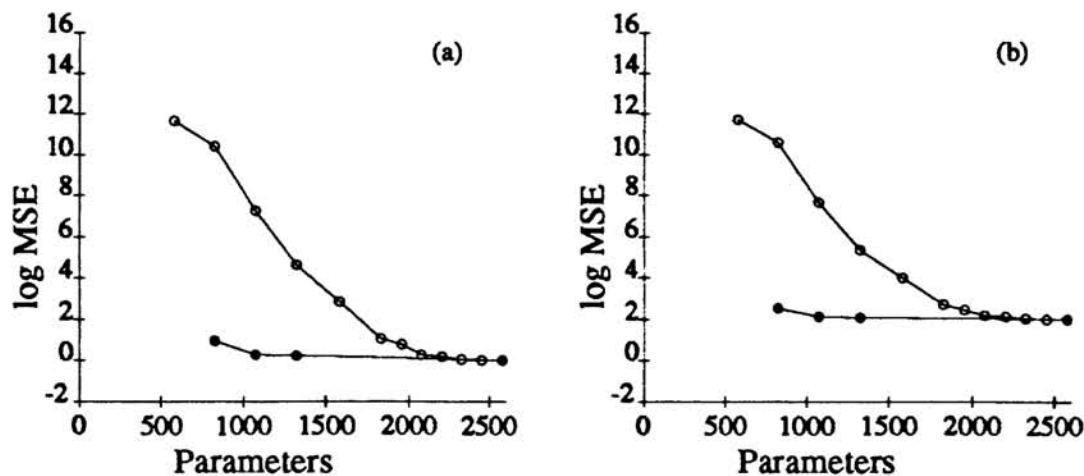

**Figure 2:** Objective function (in dB) versus number of parameters, without retraining (upper curve), and after retraining (lower curve). Curves are given for the training set (a) and the test set (b).

Figure 2 shows the log-MSE on the training set and the on the test set before and after retraining. The performance on the training set and on the test set (after retraining) stays almost the same when up to 1500 parameters (60% of the total) are deleted.

We have also used OBD as an interactive tool for network design and analysis. This contrasts with the usual view of weight deletion as a more-or-less automatic procedure. Specifically, we prepared charts depicting the saliency of the 10,000 parameters in the digit recognition network reported last year (Le Cun et al., 1990b). To our surprise, several large groups of parameters were expendable. We were able to excise the second-to-last layer, thereby reducing the number of parameters by a factor of two. The training set MSE increased by a factor of 10, and the generalization MSE increased by only 50%. The 10-category classification error on the test set actually decreased (which indicates that MSE is not the optimal

objective function for this task). OBD motivated other architectural changes, as can be seen by comparing the 2600-parameter network in (Le Cun et al., 1990a) to the 10,000-parameter network in (Le Cun et al., 1990b).

## 3  CONCLUSIONS AND OUTLOOK

We have used Optimal Brain Damage interactively to reduce the number of parameters in a practical neural network by a factor of four. We obtained an additional factor of more than two by using OBD to delete parameters automatically. The network's speed improved significantly, and its recognition accuracy increased slightly. We emphasize that the starting point was a state-of-the-art network. It would be too easy to start with a foolish network and make large improvements: a technique that can help improve an already-good network is particularly valuable.

We believe that the techniques presented here only scratch the surface of the applications where second-derivative information can and should be used. In particular, we have also been able to move beyond the approximation that "complexity equals number of free parameters" by using second-derivative information. In (Denker, Le Cun and Solla, 1990), we use it to to derive an improved measure of the network's *information content*, or complexity. This allows us to compare network architectures on a given task, and makes contact with the notion of Minimum Description Length (MDL) (Rissanen, 1989). The main idea is that a "simple" network whose description needs a small number of bits is more likely to generalize correctly than a more complex network, because it presumably has extracted the essence of the data and removed the redundancy from it.

### Acknowledgments

We thank the US Postal Service and its contractors for providing us with the database. We also thank Rich Howard and Larry Jackel for their helpful comments and encouragements. We especially thank David Rumelhart for sharing unpublished ideas.

## References

Akaike, H. (1986). Use of Statistical Models for Time Series Analysis. In *Proceedings ICASSP 86*, pages 3147–3155, Tokyo. IEEE.

Baum, E. B. and Haussler, D. (1989). What Size Net Gives Valid Generaliztion? *Neural Computation*, 1:151–160.

Becker, S. and Le Cun, Y. (1989). Improving the Convergence of Back-Propagation Learning with Second-Order Methods. In Touretzky, D., Hinton, G., and Sejnowski, T., editors, *Proc. of the 1988 Connectionist Models Summer School*, pages 29–37, San Mateo. Morgan Kaufman.

Chauvin, Y. (1989). A Back-Propagation Algorithm with Optimal Use of Hidden Units. In Touretzky, D., editor, *Neural Information Processing Systems*, volume 1, Denver, 1988. Morgan Kaufmann.

Denker, J., Schwartz, D., Wittner, B., Solla, S. A., Howard, R., Jackel, L., and Hopfield, J. (1987). Large Automatic Learning, Rule Extraction and Generalization. *Complex Systems*, 1:877–922.

Denker, J. S., Le Cun, Y., and Solla, S. A. (1990). Optimal Brain Damage. To appear in Computer and System Sciences.

Hanson, S. J. and Pratt, L. Y. (1989). Some Comparisons of Constraints for Minimal Network Construction with Back-Propagation. In Touretzky, D., editor, *Neural Information Processing Systems*, volume 1, Denver, 1988. Morgan Kaufmann.

Le Cun, Y. (1987). *Modèles Connexionnistes de l'Apprentissage*. PhD thesis, Université Pierre et Marie Curie, Paris, France.

Le Cun, Y. (1989). Generalization and Network Design Strategies. In Pfeifer, R., Schreter, Z., Fogelman, F., and Steels, L., editors, *Connectionism in Perspective*, Zurich, Switzerland. Elsevier.

Le Cun, Y., Boser, B., Denker, J. S., Henderson, D., Howard, R. E., Hubbard, W., and Jackel, L. D. (1990a). Handwritten Digit Recognition with a Back-Propagation Network. In Touretzky, D., editor, *Neural Information Processing Systems*, volume 2, Denver, 1989. Morgan Kaufman.

Le Cun, Y., Boser, B., Denker, J. S., Henderson, D., Howard, R. E., Hubbard, W., and Jackel, L. D. (1990b). Back-Propagation Applied to Handwritten Zipcode Recognition. *Neural Computation*, 1(4).

Mozer, M. C. and Smolensky, P. (1989). Skeletonization: A Technique for Trimming the Fat from a Network via Relevance Assessment. In Touretzky, D., editor, *Neural Information Processing Systems*, volume 1, Denver, 1988. Morgan Kaufmann.

Rissanen, J. (1989). *Stochastic Complexity in Statistical Inquiry*. World Scientific, Singapore.

Rumelhart, D. E. (1988). personal communication.

Solla, S. A., Schwartz, D. B., Tishby, N., and Levin, E. (1990). Supervised Learning: a Theoretical Framework. In Touretzky, D., editor, *Neural Information Processing Systems*, volume 2, Denver, 1989. Morgan Kaufman.

Vapnik, V. N. (1989). Inductive Principles of the Search for Empirical Dependences. In *Proceedings of the second annual Workshop on Computational Learning Theory*, pages 3–21. Morgan Kaufmann.

Vapnik, V. N. and Chervonenkis, A. Y. (1971). On the Uniform Convergence of Relative Frequencies of Events to Their Probabilities. *Th. Prob. and its Applications*, 17(2):264–280.

Waibel, A. (1989). Consonant Recognition by Modular Construction of Large Phonemic Time-Delay Neural Networks. In Touretzky, D., editor, *Neural Information Processing Systems*, volume 1, pages 215–223, Denver, 1988. Morgan Kaufmann.